# Generation of Internal Representation by $\alpha$-Transformation

**Ryotaro Kamimura**
Information Science Laboratory
Tokai University
1117 Kitakaname Hiratsuka Kanagawa 259-12, Japan

## Abstract

In the present paper, we propose an entropy method to transform the internal representation. The entropy function is defined with respect to the state of hidden unit, that is, internal representation. The internal representation can be transformed by changing the parameter $\alpha$ for the entropy function. Thus, the transformation is referred to as $\alpha$-transformation. The internal representation can be transformed according to given problems. By transforming the internal representation into the minimum entropy representation, we can obtain kernel networks, smaller networks with explicit interpretation. On the other hand, by changing appropriately the parameter $\alpha$, we can obtain intermediate internal representations for the improved generalization. We applied the entropy method to an autoencoder and we succeeded in obtaining kernel networks with small internal entropy. In addition, we applied the method to the frequency identification problem and we could obtain derived networks whose generalization performance was significantly superior to the performance by standard back-propagation.

## 1  Introduction

### 1.1  Creation of Internal Representation

One of the most important characteristics in the learning by neural networks is that networks can create appropriate internal representations in the course of the

learning[6]. By these internal representations, networks can solve multiple problems. However, little attention has been given to the problem, regarding what kinds of internal representations networks can create or more strongly what kind of internal representation networks should make. There have been some works [2], [4] [5] and [7] concerning the learning of the internal representation, in which the internal representation is not automatically generated. However, there has been little discussion regarding the quality or characteristics of obtained internal representations.

## 1.2   Objective

In this context, the objective of my paper is to define an entropy function for the internal representation and to formulate an entropy method to transform the internal representation according to given problems or targets, for example, the improvement of the interpretability of networks' behaviors or coding strategies, and the improvement of the generalization performance. To interpret explicitly the internal representation, and networks' behaviors, the entropy should be minimized. On the other hand, for the improved generalization performance, the entropy should appropriately be changed according to given problems.

## 1.3   Internal Entropy

Let us explain the entropy function, used in this paper. Entropy $H$ is defined with respect to the hidden unit activity,

$$H = -\sum_{i=1}^{M} p_i \log p_i, \tag{1}$$

where $p_i$ is a normalized activity of $i$th hidden unit and the summation is only over all the hidden units ($M$ hidden units). This entropy is referred to as *internal entropy*, because the entropy function is defined with respect to the internal representation. If this entropy is minimized, only one hidden unit is turned on, while all the other hidden units are turned off by multiple strong inhibitory connections [3]. On the other hand, if entropy is maximized, all the hidden units are equally activated. If entropy is sufficiently decreased, only a small number of hidden units are turned on, while all the other units are off and not used for producing outputs. Thus, this entropy function can be used to detect unnecessary hidden units to be eliminated, and to construct simple networks.

## 1.4   Transformation of Internal Representation

Let us briefly outline our ideas of the transformation of internal representations by the internal entropy. If the internal entropy is minimized, only one hidden unit is activated by an input pattern. With maximum internal entropy, all the hidden units are activated by an input pattern. Let us look at Figure 1, representing the state of internal representation. If we minimize the internal entropy, we can obtain the internal representation with minimum entropy. On the other hand, if entropy is maximized, all the hidden units are equally activated. In addition to two extreme representations, we can have multiple intermediate internal representations.

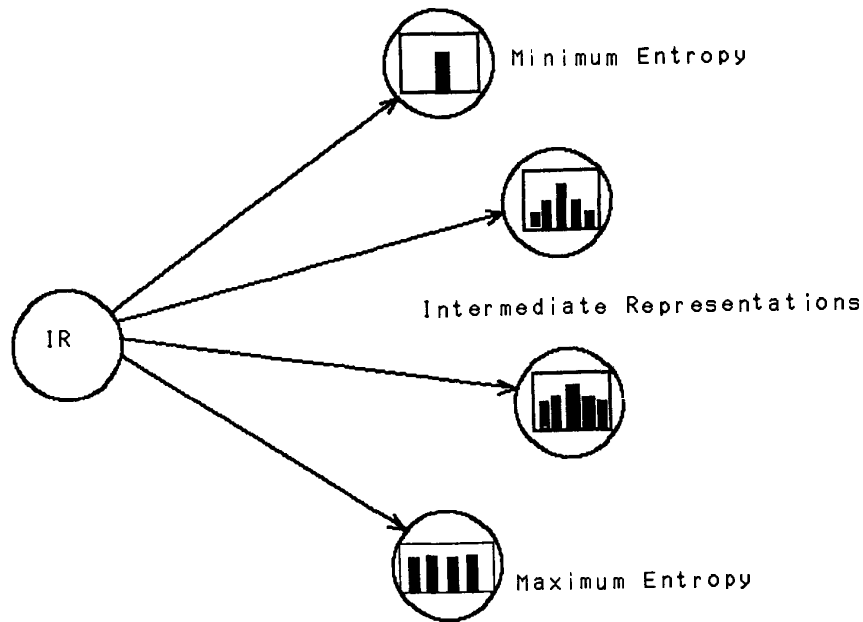

Figure 1: α-Transformation of internal representation into minimum entropy representation for the good interpretation, into maximum entropy representation and intermediate representations for the improved generalization.

For the good interpretability of the internal representation, the internal entropy should be minimized. When the internal entropy is minimized, only one hidden unit is turned on, while all the other units are off. Since only one input pattern can activate the hidden unit, it is sure that we can easily understand the meaning of the hidden unit. The hidden unit represents the information regarding the given input pattern. In a state of minimum entropy, only a small number of hidden units tend to be used. Thus, by minimizing the internal entropy, we can obtain the smallest network architecture. We call these hidden units *kernel hidden units*, and the derived network is referred to as *kernel network*. The procedure to obtain the kernel network is called *kernel hidden unit analysis*.

Concerning the generalization performance, we think that to reduce the network size is not enough to improve the generalization, as demonstrated in several experimental results, for example [1], [8]. It is necessary to adjust *appropriately* network architectures to given problems for the improved generalization performance. For example, in the case of the minimum entropy representation, the generalization performance is not expected to be improved because the minimum entropy representation tends to be a local representation and can not appropriately represent the similarity of input patterns. If we need the good generalization performance, the information concerning input patterns should be distributed over several hidden units. We think that we need intermediate representations for the improved generalization (see Figure 1).

## 2 Theory and Computational Methods

### 2.1 Entropy Method

We have applied entropy minimization method to recurrent back-propagation[3]. In this section, we formulate the entropy method for standard back-propagation. Suppose that a network is composed of three layers: input, competitive hidden and output layers. Hidden units are denoted by $v_i$ and input terminals by $\xi_j$. Then, connections from inputs to hidden units are denoted by $w_{ij}$ and connections from hidden units to output units are denoted by $W_{ij}$. A hidden unit produces an output

$$v_i = f(u_i),$$

where

$$u_i = \sum_{j=1}^{N} w_{ij}\xi_j.$$

where $\xi_i$ is a $i$th element of an input pattern, $N$ is the number of elements in the pattern and $f$ is the sigmoid function, defined by

$$f(u_i) = \frac{1}{1 + e^{-u_i}}.$$

An entropy function on competitive hidden layer is defined by

$$H = -\sum_{i=1}^{M} p_i \log p_i, \tag{2}$$

where

$$p_i = \frac{v_i}{\sum_{r}^{M} v_r},$$

and $M$ is the number of competitive hidden units.

Differentiating entropy function with respect to connections from input to hidden layer, we have

$$
\begin{aligned}
-\frac{\partial H}{\partial w_{ij}} &= -\frac{\partial H}{\partial v_i}\frac{\partial v_i}{\partial w_{ij}} \\
&= \phi_i \xi_j, \tag{3}
\end{aligned}
$$

where

$$\phi_i = (\log p_i + 1)p_i(1 - p_i)(1 - v_i). \tag{4}$$

By using this $\phi$ function, update rules can be summarized as follows. First, for connections from competitive hidden units to output units, only delta rule must be used. For connections from input units to competitive hidden units, in addition to delta rule, the $\phi$ function must be incorporated as

$$
\begin{aligned}
\Delta w_{ij} &= -\alpha \frac{\partial H}{\partial w_{ij}} - \beta \frac{\partial E}{\partial w_{ij}} \\
&= \alpha \phi_i \xi_j + \beta \delta_i \xi_j. \tag{5}
\end{aligned}
$$

This update rule means that in addition to the error minimization, entropy must be minimized or maximized in the course of the learning by changing the parameter $\alpha$.

## 2.2 Kernel Hidden Unit Analysis

As already mentioned, we have used the entropy method for obtaining simple networks, called *kernel networks*. Let us briefly explain the procedure of kernel hidden unit analysis. We have observed that by minimizing entropy, we can obtain small network architectures, compared with original oversized network architecture. In a state of minimum entropy, a smaller number of hidden units tended to be used to produced targets. We call these units *kernel hidden units*. To determine kernel hidden units, we have introduced the variance of input-hidden connections. The variance of input-hidden connections is used to measure how a given hidden units is important for producing targets correctly. This variance is used because it has extensively been observed in our experiments that hidden units playing the important roles tend to have the large variance, compared with other unimportant hidden units. The variance $(s_i^2)$ of $i$th hidden unit is defined by

$$s_i^2 = \frac{1}{M-1} \sum_{j=1}^{M} (w_{ij} - \overline{w_i})^2,$$

where $M$ is the number of hidden units, $\overline{w_i}$ is an average over all the connections into $i$th hidden units. With these kernel hidden units, small networks, called *kernel networks*, can be obtained, whose performance with respect to the error minimization is completely equivalent to the original networks with a large number of hidden units. Moreover, we can easily interpret the behaviors of kernel networks, because the network size is small. Finally, this process of obtaining a small network architecture is just the *kernel hidden unit analysis*.

## 3 Results

### 3.1 Transformation into Kernel Networks

We applied the method to a network in which thirty-five input, hidden and output units were employed. The network must exactly reproduce five alphabet letters: $B$, $C$, $D$, $E$, $F$, $G$ at output units. Since the difference between these letters are small, compared with the difference between other letters, these letters are expected to be compressed into a smaller number of hidden units.

A minimum entropy was searched by changing the parameter $\alpha$. Entropy was decreased gradually as the parameter was increased as shown in Figure 2. By using kernel hidden unit analysis, we observed only three major hidden units for entropy method. Thus, a kernel network is composed of three kernel hidden units, compared with thirty-five hidden units of the original network. On the other hand, by using standard back-propagation, many hidden units are activated, and thus the information upon input patterns are distributed over many hidden units. Figure 3 shows original network and kernel network for the autoencoder. As you can see from the figure, the number of hidden units is decreased from thirty-five to three by using the kernel hidden unit analysis.

Finally, to see clearly the meaning of hidden units, networks were constructed only with kernel hidden units, and the outputs, generated by the networks were carefully examined. We could see that the role of hidden units could explicitly be determined.

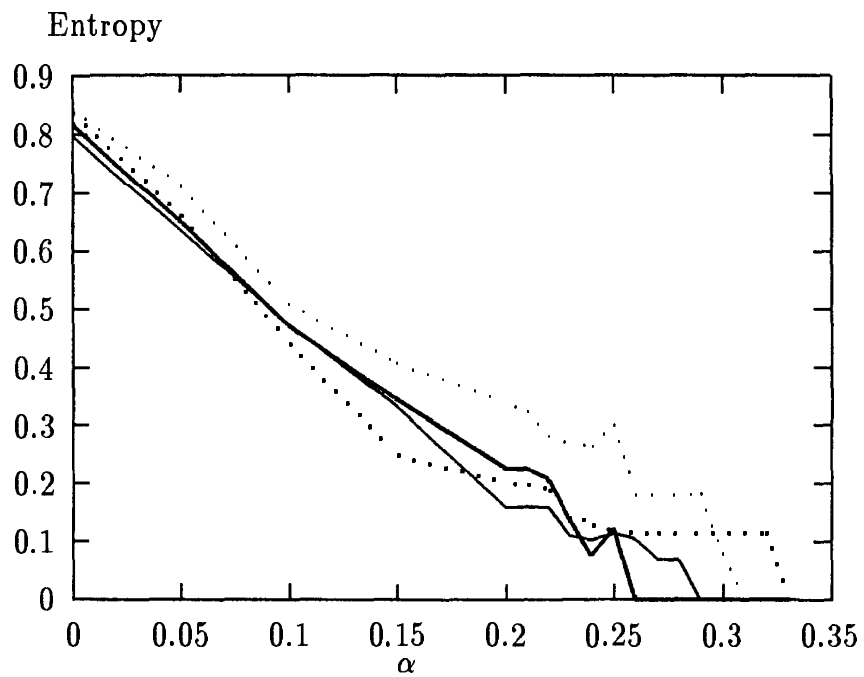

Figure 2: Entropies, computed with four different initial values as a function of the parameter $\alpha$ for an autoencoder.

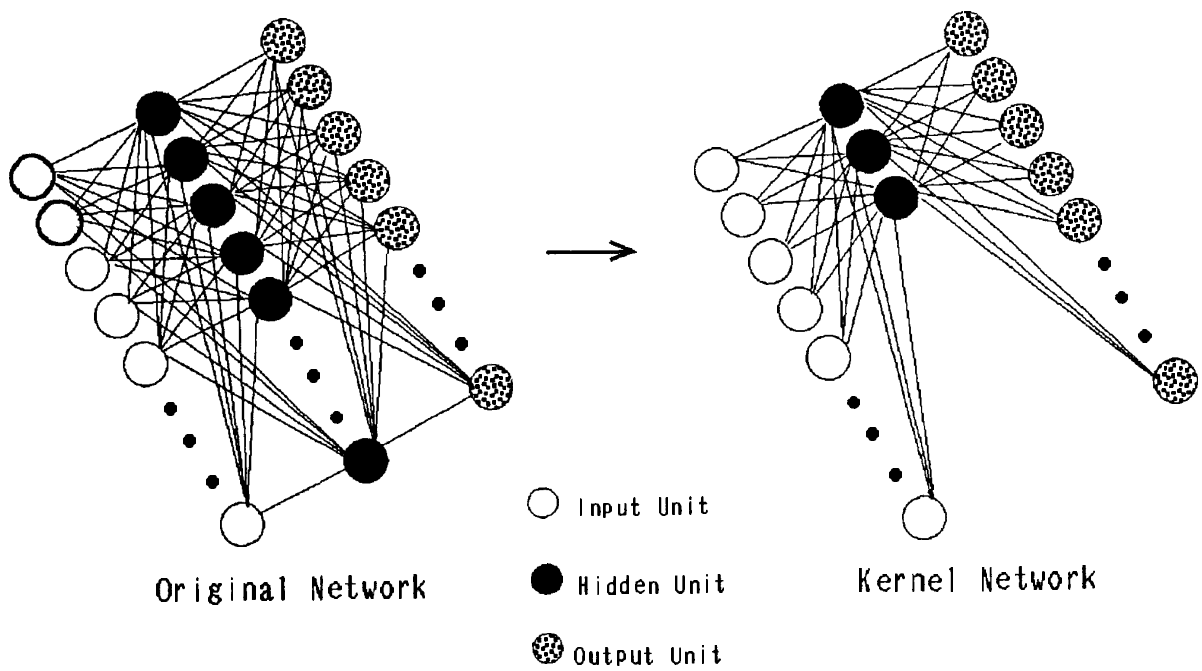

Figure 3: Original network and kernel network for the autoencoder.

Table 1: Summary of experiments for networks with fifteen hidden units. Values in this table were averages over ten initial conditions.

| $\alpha(\times 10^{-3})$ | Entropy | HD(%) | SSE |
|---|---|---|---|
| 0 | 0.623 | 6.296 | 4.118 |
| -2.031 | 0.778 | 1.111 | |
| -3.250 | 0.785 | | 1.687 |

## 3.2 Transformation for the Improved Generalization

As already mentioned, we think that the generalization can be improved by adjusting network architectures to given problems appropriately. In this section, we try to change the parameter $\alpha$ and thus to change the internal entropy to obtain the improved generalization performance. Our main result, presented in this section is that the generalization performance can be improved by increasing the entropy function.

We applied our method to the identification of frequencies[8]. Networks must identify three frequencies of sine waves with phases, different from those of training data sets. First, training data were divided into three classes with three different frequencies (2, 4, 6). Each class has ten exemplars with sixty-four samples from sine waves. Thus, the number of input units was sixty-four. The number of output units was three and specific target values were assigned to each output unit, according to the frequency of a class. Experiments were performed with different initial values for weights, ranging between -0.5 and 0.5. The learning was considered to be finished when the absolute differences between targets and outputs were all below 0.1. Finally, the learning rate $\beta$ was set to 0.1 for all the experiments.

Let us see Table 1, showing the summary of experimental results with ten and fifteen units by ten different initial conditions. Values in the second row in the table could give the lowest values of Hamming distance. In the same way, values in the third row could give the lowest SSE. The number of hidden units was fifteen. As you can see from the table, when entropy was increased from 0.623 to 0.778, Hamming distance was decreased greatly from 6.296 to 1.111. In addition, five out of ten initial conditions could produce zero Hamming distance, meaning that the generalization performance was perfect. By the criterion of SSE, when entropy was increased from 0.623 to 0.785, SSE was decreased from 4.118 to 1.687. All the results are statistical significant. These results show that by using entropy method, the generalization performance was significantly improved.

## 4    Conclusion

In this paper, we have proposed an entropy method to transform the internal representation. The internal entropy has been defined with respect to the hidden unit activity or the internal representation. If the internal entropy is minimized, only one hidden unit tends to respond to a specific input pattern, thus, the internal representation is as local as possible. In this case, it is easy to understand the meaning or the function of the hidden unit, because the hidden unit tends to respond to

only one specific input pattern. However, we have observed that in the case of the minimum entropy representation, the generalization performance is not improved. To obtain the improved generalization performance, the information regarding an input pattern should be distributed over several hidden units. We have applied the entropy method to an autoencoder. In this problem, we have observed that by minimizing the internal entropy, we can obtain smaller networks, called *kernel networks*. These kernel networks are so small that we can easily understand the meaning of the internal representation or networks' behaviors. Then, we have attempted to change the internal entropy for the improve generalization. We have observed that up to a certain point, the generalization performance can be improved by increasing the internal entropy. Finally, if relations between kernel and intermediate networks can explicitly be determined, we can obtain networks with the good interpretability and the improved generalization performance.

# References

[1] Y. Chauvin, "A backpropagation algorithm with optimal use of hidden units," in *Advances in Neural Information Processing Systems*, D. S. Touretzky, Eds, San Mateo: CA, pp.519-526, 1989.

[2] T. Grossman, "The CHIR algorithm for feed forward networks with binary weights," in *Advances in Neural Information Processing Systems*, D. S. Touretzky, Eds, Vol.3, San Mateo: CA, pp.516-523, 1991.

[3] R. Kamimura, "Minimum entropy method in neural networks," in *Proceeding of 1993 IEEE International Conference on Neural Networks*, Vol.1, pp.219-225, 1993.

[4] A. Kroph, G. I. Thorbegsson and J. A. Hertz, "A cost function of internal representation," in *Advances in Neural Information Processing Systems*, D. S. Touretzky, Eds, Vol.3, San Mateo: CA, pp.733-740, 1991.

[5] R. Rohwer, "The moving target training algorithm," in *Advances in Neural Information Processing Systems*, D. S. Touretzky, Eds, Vol.3, San Mateo: CA, pp.558-565, 1991.

[6] D. E. Rumelhart, G. E. Hinton and R. J. Williams, "Learning internal representation by error propagation," in *Parallel Distributed Processing*, D. E. Rumelhart, J. L. McClelland, and the PDP Research Group, Cambridge, Massachusetts: the MIT Press, Vol.1, pp.318-362, 1986.

[7] D. Saad and E. Marom, "Learning by choice of internal representation: an energy minimization approach," *Complex Systems*, Vol.4, pp.107-118, 1990.

[8] J. Sietsma and R. J. F. Dow, "Creating artificial neural networks that generalize," *Neural Networks*, Vol.4, pp.67-79, 1991.